# Speaker Independent Speech Recognition with Neural Networks and Speech Knowledge

**Yoshua Bengio**
Dept Computer Science
McGill University
Montreal, Canada H3A2A7

**Renato De Mori**
Dept Computer Science
McGill University

**Regis Cardin**
Dept Computer Science
McGill University

## ABSTRACT

We attempt to combine neural networks with knowledge from speech science to build a speaker independent speech recognition system. This knowledge is utilized in designing the preprocessing, input coding, output coding, output supervision and architectural constraints. To handle the temporal aspect of speech we combine delays, copies of activations of hidden and output units at the input level, and Back-Propagation for Sequences (BPS), a learning algorithm for networks with local self-loops. This strategy is demonstrated in several experiments, in particular a nasal discrimination task for which the application of a speech theory hypothesis dramatically improved generalization.

## 1  INTRODUCTION

The strategy put forward in this research effort is to combine the flexibility and learning abilities of neural networks with as much knowledge from speech science as possible in order to build a speaker independent automatic speech recognition system. This knowledge is utilized in each of the steps in the construction of an automated speech recognition system: preprocessing, input coding, output coding, output supervision, architectural design. In particular

for preprocessing we explored the advantages of various possible ways of processing the speech signal, such as comparing an ear model *vs.* Fast Fourier Transform (FFT), or compressing the frame sequence in such a way as to conserve an approximately constant rate of change. To handle the temporal aspect of speech we propose to combine various algorithms depending of the demands of the task, including an algorithm for a type of recurrent network which includes only self-loops and is local in space and time (BPS). This strategy is demonstrated in several experiments, in particular a nasal discrimination task for which the application of a speech theory hypothesis drastically improved generalization.

## 2  Application of Speech Knowledge

### 2.1  Preprocessing

Our previous work has shown us that the choice of preprocessing significantly influences the performance of a neural network recognizer. (*e.g.*, Bengio & De Mori 1988) Different types of preprocessing processes and acoustic features can be utilized at the input of a neural network. We used several acoustic features (such as counts of zero crossings), filters derived from the FFT, energy levels (of both the signal and its derivative) and ratios (Gori, Bengio & De Mori 1989), as well as an ear model and synchrony detector.

#### Ear model *vs.* FFT

We performed experiments in speaker-independent recognition of 10 english vowels on isolated words that compared the use of an ear model with an FFT as preprocessing. The FFT was done using a mel scale and the same number of filters (40) as for the ear model. The ear model was derived from the one proposed by Seneff (1985). Recognition was performed with a neural network with one hidden layer of 20 units. We obtained 87% recognition with the FFT preprocessing *vs.* 96% recognition with the ear model (plus synchrony detector to extract spectral regularity from the instantaneous output of the ear model)(Bengio, Cosi, De Mori 1989). This was an example of the successful application of knowledge about human audition to the automatic recognition of speech with machines.

#### Compression in time resulting in constant rate of change

The motivation for this processing step is the following. The rate of change of the speech signal, (as well as the output of networks performing acoustic→phonetic mappings) varies a lot. It would be nice to have more temporal precision in parts of the signal where there is a lot of variation (bursts, fast transitions) and less temporal precision in more stable parts of the signal (e.g., vowels, silence).

Given a sequence of vectors (parameters, which can be acoustic parameters, such as spectral coefficients, as well as outputs from neural networks) we transform it by compressing it in time in order to obtain a shorter sequence where frames refer to segments of varying length of the original sequence.

Very simple Algorithm that *maps sequence X(t) → sequence Y(t)* where X and Y are vectors:

```
{ Accumulate and average X(t), X(t+1)...X(t+n) in Y(s) as
long as the sum of the Distance(X(t),X(t+1)) + ... +
Distance(X(t+n-1),X(t+n)) is less than a threshold.
When this threshold is reached,
  t←t+n+1;
  s←s+1; }
```

The advantages of this system are the following: 1) more temporal precision where needed, 2) reduction of the dimensionality of the problem, 3) constant rate of change of the resulting signal so that when using input windows in a neural net, the windows may have less frames, 4) better generalization since several realizations of the same word spoken at different rates of speech tend to be reduced to more similar sequences.

Initial results when this system is used to compress spectral parameters (24 mel-scaled FFT filters + energy) computed every 5 ms were interesting. The task was the classification of phonemes into 14 classes. The size of the database was reduced by 30%. The size of the window was reduced (4 frames instead of 8), hence the network size was reduced as well. Half the size of the window was necessary in order to obtain similar performance on the training set. Generalization on the test set was slightly better (from 38% to 33% classification error by frame). The idea to use a measure of rate of change to process speech is not new (Atal, 1983) but we believe that it might be particularly useful when the recognition device is a neural network with an input of several frames of acoustic parameters.

## 2.2   Input coding

Our previous work has shown us that information should be as easily accessible as possible to the network. For example, compression of the spectral information into cepstrum coefficients (with first few coefficients having very large variance) resulted in poorer performance with respect to experiments done with the spectrum itself. The recognition was performed with a neural network where units compute the sigmoid of the weighted sum of their inputs. The task was the broad classification of phonemes in 4 classes. The error on the test set increased from 15% to 20% when using cepstral rather than spectral coefficients.

Another example concerns the recognition experiments for which there is a lot of variance in the quantities presented in the input. A grid representation with coarse coding improved learning time as well as generalization (since the problem became more separable and thus the network needed less hidden units). (Bengio, De Mori, 1988).

## 2.3   Output coding

We have chosen an output coding scheme based on phonetic features defined by the way speech is produced. This is generally more difficult to learn but results in better generalization, especially with respect to new sounds that had

not been seen by the network during the training. We have demonstrated this with experiments on vowel recognition in which the networks were trained to recognized the place and the manner of articulation (Bengio, Cosi, De Mori 89). In addition the resulting representation is more compact than when using one output for each phoneme. However, this representation remains meaningful i.e. each output can be attributed a meaning almost independently of the values of the other outputs.

In general, an explicit representation is preferred to an arbitrary and compact one (such as a compact binary coding of the classes). Otherwise, the network must perform an additional step of encoding. This can be costly in terms of the size of the networks, and generally also in terms of generalization (given the need for a larger number of weights).

## 2.4    Output supervision

When using a network with some recurrences it is not necessary that supervision be provided at every frame for every output (particularly for transition periods which are difficult to label). Instead the supervision should be provided to the network when the speech signal clearly corresponds to the categories one is trying to learn. We have used this approach when performing the discrimination between /b/ and /d/ with the BPS (Back Propagation for Sequences) algorithm (self-loop only, *c.f.* section 3.3).

Giving additional information to the network through more supervision (with extra output units) improved learning time and generalization ( *c.f.* section 4).

## 2.5    Architectural design

Hypothesis about the nature of the processing to be performed by the network based on speech science knowledge enables to put constraints on the architecture. These constraints result in a network that generalizes better than a fully connected network. This strategy is most useful when the speech recognition task has been modularized in the appropriate way so that the same architectural constraints do not have to apply to all of the subtasks. Here are several examples of application of modularization. We initially explored modularization by acoustic context (different networks are triggered when various acoustic contexts are detected)(Bengio, Cardin, De Mori, Merlo 89) We also implemented modularisation by independent articulatory features (vertical and horizontal place of articulation) (in Bengio, Cosi, De Mori, 89). Another type of modularization, by subsets of phonemes, was explored by several researchers, in particular Alex Waibel (Waibel 88).

# 3   Temporal aspect of the speech recognition task

Both of the algorithms presented in the following subsections assume that one is using the Least Mean Square Error criterion, but both can be easily modified for any type of error criterion. We used and sometimes combined the following techniques:

### 3.1    Delays

If the speech signal is preprocessed in such a way as to obtain a frame of acoustic parameters for every interval of time, one can use delays from the input units representing these acoustic parameters to implement an input window on the input sequence, as in NETtalk, or using this strategy at every level as in TDNNs (Waibel 88). Even when we use a recurrent network, a small number of delays on the outgoing links of the input units might be useful. It enables the network to make a direct comparison between successive frames.

### 3.2    BPS (Back Propagation for Sequences)

This is a learning algorithm that we have introduced for networks that have a certain constrained type of recurrence (local self-loops). It permits to compute the gradient of the error with respect to all weights. This algorithm has the same order of space and time requirements as backpropagation for feedforward networks. Experiments with the /b/ *vs.* /d/ speaker independent discrimination yielded 3.45% error on the test set for the BPS network as opposed to 6.9% error for a feedforward network (Gori, Bengio, De Mori 89).

BPS equations:

*feedforward pass:*

●dynamic units: these have a local self-loop and their input must directly come from the input layer.

$$X_i(t+1) = W_{ii} X_i(t) + \sum_j W_{ij} f(X_j(t))$$

$$\partial X_i(t+1)/\partial W_{ij} = W_{ii} \partial X_i(t)/\partial W_{ij} + f(X_j(t)) \qquad \text{for } i\mathrel{!}{=}j$$

$$\partial X_i(t)/\partial W_{ii} = W_{ii} \partial X_i(t)/\partial W_{ii} + X_i(t) \qquad \text{for } i{=}{=}j$$

●static units, *i.e.*, without feedback, follow usual Back-Propagation (BP) equations (Rumelhart et al. 1986):

$$X_i(t+1) = \sum_j W_{ij} f(X_j(t))$$

$$\partial X_i(t+1)/\partial W_{ij} = f(X_j(t))$$

*Backpropagation pass*, after every frame: as usual but using above definition of $\partial X_i(t)/\partial W_{ii}$ instead of the usual $f(X_j(t))$.

This algorithm has a time complexity $O(L \cdot N_w)$(as static BP) It needs space $O(N_u)$, where L is the length of a sequence, $N_w$ is the number of weights and $N_u$ is the number of units. Note that it is local in time (it is causal, no backpropagation in time) and in space (only information coming from direct neighbors is needed).

### 3.3    Discrete Recurrent Net without Constraints

This is how we compute the gradient in an unconstrained discrete recurrent net. The derivation is similar to the one of Pearlmutter (1989). It is another way to view the computation of the gradient for recurrent networks, called time unfolding, which was presented by (Rumelhart et al. 1986). Here the units have a memory of their past activations during the forward pass (from

frame 1 to L) and a "memory" of the future $\partial E/\partial Xi$ during the backward pass (from frame L down to frame 1).

*Forward phase:* consider the possibility of an arbitrary number of connections from unit i to unit j, each having a different delay d.

$$Xi(t) = \sum_{j,d} Wijd \ f(Xi(t-d)) + I(i,t)$$

Here, the basic idea is to compute $\partial E/\partial Wijd$ by computing $\partial E/\partial Xi(t)$:

$$\partial E/\partial Wijd = \sum_t \partial E/\partial Xi(t) \ \partial Xi(t)/\partial Wijd$$

where $\partial Xi(t)/\partial Wijd = f(Xj(t-d))$ as usual. In the *backward phase* we backpropagate $\partial E/\partial Xi(t)$ recursively from the last time frame=L down to frame 1:

$$\partial E/\partial Xi(t) = \sum_{k,d} Wkid \ \partial E/\partial Xk(t+d) \ f'(Xj(t))$$
$$+(\text{if } i \text{ is an output unit})(f(Xi(t))-Yi^*(t)) \ f'(Xi(t))$$

where $Yi^*(t)$ is the target output for unit i at time t. In this equation the first term represents backpropagation from future times and downstream units, while the second one comes from direct external supervision. This algorithm works for any connectivity of the recurrent network with delays. Its time complexity is $O(L \cdot Nw)$ (as static BP). However the space requirements are $O(L \cdot Nu)$. The algorithm is local in space but not in time; however, we found that restriction not to be very important in speech recognition, where we consider at most a few hundred frames of left context (one sentence).

## 4  Nasal experiment

As an example of the application of the above described strategy we have performed the following experiment with the discrimination of nasals /m/ and /n/ in a fixed context. The speech material consisted of 294 tokens from 70 training speakers (male and female with various accents) and 38 tokens from 10 test speakers. The speech signal is preprocessed with an ear model followed by a generalized synchrony detector yielding 40 spectral parameters every 10 ms. Early experiments with a simple output coding {vowel, m, n}, a window of two consecutive frames as input, and a two-layer fully connected architecture with 10 hidden units gave poor results: 15% error on the test set. A speech theory hypothesis claiming that the most critical discriminatory information for the nasals is available during the transition between the vowel and the nasal inspired us to try the following output coding: {vowel, transition to m, transition to n, nasal}. Since the transition was more important we chose as input a window of 4 frames at times t, t-10ms, t-30ms and t-70ms. To reduce the connectivity the architecture included a constrained first hidden layer of 40 units where each unit was meant to correspond to one of the 40 spectral frequencies of the preprocessing stage. Each such hidden unit associated with filter bank F was connected (when possible) to input units corresponding to

frequency banks      (F-2,F-1,F,F+1,F+2)

and times            (t,t-10ms,t-30ms,t-70ms).

Experiments with this feedforward delay network (160 inputs--40 hidden--10 hidden--4 outputs) showed that, indeed the strongest clues about the identity of the nasal seemed to be available during the transition and for a very short time, just before the steady part of the nasal started. In order to extract that critical information from the stream of outputs of this network, a second network was trained on the outputs of the first one to provide clearly the discrimination of the nasal during the whole of the nasal. That higher level network used the BPS algorithm to learn about the temporal nature of the task and keep the detected critical information during the length of the nasal. Recognition performance reached a plateau of 1.14% errors on the training set. Generalization was very good with only 2.63% error on the test set.

## 5  Future experiments

One of the advantages of using phonetic features instead of phonemes to describe the speech is that they could help to learn more robustly about the influence of context. If one uses a phonemic representation and tries to characterize the influence of the past phoneme on the current phoneme, one faces the problem of poor statistical sampling of many of the corresponding diphones (in a realistic database). On the other hand, if speech is characterized by several independent dimensions such as horizontal and vertical place of articulation and voicing, then the number of possible contexts to consider for each value of one of the dimensions is much more limited. Hence the set of examples characterizing those contexts is much richer.

We now present some observations on continuous speech based on our initial work with the TIMIT database in which we try learning articulatory features. Although we have obtained good results for the recognition of articulatory features (horizontal and vertical place of articulation) for isolated words, initial results with continuous speech are less encouraging. Indeed, whereas the measured place of articulation (by the networks) for phonemes in isolated speech corresponds well to expectations (as defined by acousticians who physically measured these features for isolated short words), this is not the case for continuous speech. In the latter case, phonemes have a much shorter duration so that the articulatory features are most of the time in transition, and the place of articulation generally does not reach the expected target values (although it always moves in the right *direction* ). This is probably due to the inertia of the production system and to coarticulation effects. In order to attack that problem we intend to perform the following experiments. We could use the subset of the database for which the phoneme duration is sufficiently long to learn an approximation of the articulatory features. We could then improve that approximation in order to be able to learn about the trajectories of these features found in the transitions from one phoneme to the next. This could be done by using a two stage network (similar to the encoder network) with a bottleneck in the middle. The first stage of the network produces phonetic features and receives supervision only on the steady parts of the speech. The second stage of the network (which would be a recurrent network) has as input the trajectory of the approximation of the phonetic features and produces as output the previous, current and next phoneme. As an additional constraint, we propose to use self-loops with various time constants on the units of the bottleneck. Units that represent fast varying descrip-

tors of speech will have a short time constant, while units that we want to have represent information about the past acoustic context will have a slightly longer time constant and units that could represent very long time range information — such as information about the speaker or the recording conditions — will receive a very long time constant.

This paper has proposed a general strategy for setting up a speaker independent speech recognition system with neural networks using as much speech knowledge as possible. We explored several aspects of this problem including preprocessing, input coding, output coding, output supervision, architectural design, algorithms for recurrent networks, and have described several initial experimental results to support these ideas.

### References

Atal B.S. (1983), Efficient coding of LPC parameters by temporal decomposition, *Proc. ICASSP 83* , Boston, pp 81-84.

Bengio Y., Cardin R., De Mori R., Merlo E. (1989) Programmable execution of multi-layered networks for automatic speech recognition, *Communications of the Association for Computing Machinery,* 32 (2).

Bengio Y., Cardin R., De Mori R., (1990), Speaker independent speech recognition with neural networks and speech knowledge, in D.S. Touretzky (ed.), *Advances in Neural Networks Information Processing Systems 2,* San Mateo, CA: Morgan Kaufmann.

Bengio Y., De Mori R., (1988), Speaker normalization and automatic speech recognition using spectral lines and neural networks, *Proc. Canadian Conference on Artificial Intelligence (CSCSI-88)* , Edmonton Al., May 88.

Bengio Y., Cosi P., De Mori R., (1989), On the generalization capability of multi-layered networks in the extraction of speech properties, *Proc. Internation Joint Conference of Artificial Intelligence (IJCAI 89)"* , Detroit, August 89, pp. 1531-1536.

Gori M., Bengio Y., De Mori R., (1989), BPS: a learning algorithm for capturing the dynamic nature of speech, *Proc. IEEE International Joint Conference on Neural Networks,* Washington, June 89.

Pearlmutter B.A., Learning state space trajectories in recurrent neural networks, (1989), *Neural Computation, vol. 1, no. 2,* pp. 263-269.

Rumelhart D.E., Hinton G., Williams R.J., (1986), Learning internal representation by error propagation, in *Parallel Distributed Processing, exploration in the microstructure of cognition, vol. 1,* MIT Press 1986.

Seneff S., (1985), Pitch and spectral analysis of speech based on an auditory synchrony model, *RLE Technical report 504,* MIT.

Waibel A., (1988), Modularity in neural networks for speech recognition, *Advances in Neural Networks Information Processing Systems 1.* San Mateo, CA: Morgan Kaufmann.
